# Distributed Synchrony of Spiking Neurons in a Hebbian Cell Assembly

**David Horn** **Nir Levy**
School of Physics and Astronomy,
Raymond and Beverly Sackler Faculty of Exact Sciences,
Tel Aviv University, Tel Aviv 69978, Israel
horn@neuron.tau.ac.il    nirlevy@post.tau.ac.il

**Isaac Meilijson** **Eytan Ruppin**
School of Mathematical Sciences,
Raymond and Beverly Sackler Faculty of Exact Sciences,
Tel Aviv University, Tel Aviv 69978, Israel
isaco@math.tau.ac.il    ruppin@math.tau.ac.il

## Abstract

We investigate the behavior of a Hebbian cell assembly of spiking neurons formed via a temporal synaptic learning curve. This learning function is based on recent experimental findings. It includes potentiation for short time delays between pre- and post-synaptic neuronal spiking, and depression for spiking events occuring in the reverse order. The coupling between the dynamics of the synaptic learning and of the neuronal activation leads to interesting results. We find that the cell assembly can fire asynchronously, but may also function in complete synchrony, or in distributed synchrony. The latter implies spontaneous division of the Hebbian cell assembly into groups of cells that fire in a cyclic manner. We invetigate the behavior of distributed synchrony both by simulations and by analytic calculations of the resulting synaptic distributions.

## 1   Introduction

The Hebbian paradigm that serves as the basis for models of associative memory is often conceived as the statement that a group of excitatory neurons (the Hebbian cell assembly) that are coupled synaptically to one another fire together when a subset of the group is being excited by an external input. Yet the details of the temporal spiking patterns of neurons in such an assembly are still ill understood. Theoretically it seems quite obvious that there are two general types of behavior: synchronous neuronal firing, and asynchrony where no temporal order exists in the assembly and the different neurons fire randomly but with the same overall rate. Further subclassifications were recently suggested by [Brunel, 1999]. Experimentally this question is far from being settled because evidence for the associative

memory paradigm is quite scarce. On one hand, one possible realization of associative memories in the brain was demonstrated by [Miyashita, 1988] in the inferotemporal cortex. This area was recently reinvestigated by [Yakovlev et al., 1998] who compared their experimental results with a model of asynchronized spiking neurons. On the other hand there exists experimental evidence [Abeles, 1982] for temporal activity patterns in the frontal cortex that Abeles called synfire-chains. Could they correspond to an alternative type of synchronous realization of a memory attractor?

To answer these questions and study the possible realizations of attractors in cortical-like networks we investigate the temporal structure of an attractor assuming the existence of a synaptic learning curve that is continuously applied to the memory system. This learning curve is motivated by the experimental observations of [Markram et al., 1997, Zhang et al., 1998] that synaptic potentiation or depression occurs within a critical time window in which both pre- and post-synaptic neurons have to fire. If the pre-synaptic neuron fires first within 30ms or so, potentiation will take place. Depression is the rule for the reverse order.

The regulatory effects of such a synaptic learning curve on the synapses of *a single neuron* that is subjected to external inputs were investigated by [Abbott and Song, 1999] and by [Kempter et al., 1999]. We investigate here the effect of such a rule within an *assembly of neurons* that are all excited by the same external input throughout a training period, and are allowed to influence one another through their resulting sustained activity.

## 2    The Model

We study a network composed of $N_E$ excitatory and $N_I$ inhibitory integrate-and-fire neurons. Each neuron in the network is described by its subthreshold membrane potential $V_i(t)$ obeying

$$\dot{V}_i(t) = -\frac{1}{\tau_n}V_i(t) + RI_i(t) \tag{1}$$

where $\tau_n$ is the neuronal integration time constant. A spike is generated when $V_i(t)$ reaches the threshold $V_{rest} + \theta$, upon which a refractory period of $\tau_{RP}$ is set on and the membrane potential is reset to $V_{reset}$ where $V_{rest} < V_{reset} < V_{rest} + \theta$. $I_i(t)$ is the sum of recurrent and external synaptic current inputs. The net synaptic input charging the membrane of excitatory neuron $i$ at time $t$ is

$$RI_i(t) = \sum_j J_{ij}^{EE}(t) \sum_l \delta\left(t - t_j^l - \tau_d\right) - \sum_j J_{ij}^{EI} \sum_m \delta\left(t - t_j^m - \tau_d\right) + I^{ext} \tag{2}$$

summing over the different synapses of $j = 1, \ldots, N_E$ excitatory neurons and of $j = 1, \ldots, N_I$ inhibitory neurons, with postsynaptic efficacies $J_{ij}^{EE}(t)$ and $J_{ij}^{EI}$ respectively. The sum over $l$ ($m$) represents a sum on different spikes arriving at synapse $j$, at times $t = t_j^l + \tau_d$ ($t = t_j^m + \tau_d$), where $t_j^l$ ($t_j^m$) is the emission time of the $l$-th ($m$-th) spike from the excitatory (inhibitory) neuron $j$ and $\tau_d$ is the synaptic delay. $I^{ext}$, the external current, is assumed to be random and independent at each neuron and each time step, drawn from a Poisson distribution with mean $\lambda^{ext}$. Analogously, the synaptic input to the inhibitory neuron $i$ at time $t$ is

$$RI_i(t) = \sum_j J_{ij}^{IE} \sum_l \delta\left(t - t_j^l - \tau_d\right) - \sum_j J_{ij}^{II} \sum_m \delta\left(t - t_j^m - \tau_d\right) + I^{ext}. \tag{3}$$

We assume full connectivity among the excitatory neurons, but only partial connectivity between all other three types of possible connnections, with connection

probabilities denoted by $C^{EI}$, $C^{IE}$ and $C^{II}$. In the following we will report simulation results in which the synaptic delays $\tau_d$ were assigned to each synapse, or pair of neurons, randomly, chosen from some finite set of values. Our analytic calculation will be done for one fixed value of this delay parameter.

The synaptic efficacies between excitatory neurons are assumed to be potentiated or depressed according to the firing patterns of the pre- and post-synaptic neurons. In addition we allow for a uniform synaptic decay. Thus each excitatory synapse obeys

$$\dot{J}_{ij}^{EE}(t) = -\frac{1}{\tau_s} J_{ij}^{EE}(t) + F_{ij}(t) \tag{4}$$

where the synaptic decay constant $\tau_s$ is assumed to be very large compared to the membrane time constant $\tau_n$. $J_{ij}^{EE}(t)$ are constrained to vary in the range $[0, J_{max}]$. The change in synaptic efficacy is defined by $F_{ij}(t)$, as

$$F_{ij}(t) = \sum_{k,l} \left[ \delta(t - t_i^k) K_P(t_j^l - t_i^k) + \delta(t - t_j^l) K_D(t_j^l - t_i^k) \right] \tag{5}$$

where $K_P$ and $K_D$ are the potentiation and depression branches of the kernel function

$$K(\delta) = -c\delta \exp \left[ -(a\delta + b)^2 \right] \tag{6}$$

plotted in Figure 1. Following [Zhang *et al.*, 1998] we distinguish between the situation where the postsynaptic spike, at $t_i^k$, appears after or before the presynaptic spike, at $t_j^l$, using the asymmetric kernel that captures the essence of their experimental observations.

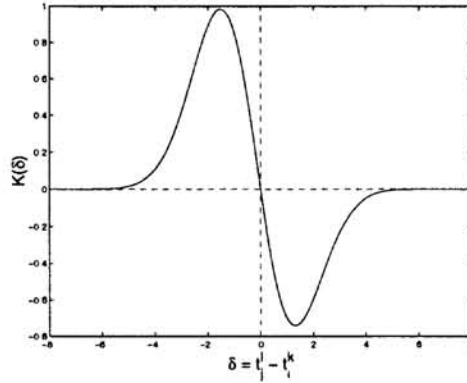

Figure 1: The kernel function whose left part, $K_P$, leads to potentiation of the synapse, and whose right branch, $K_D$, causes synaptic depression.

## 3  Distributed Synchrony of a Hebbian Assembly

We have run our system with synaptic delays chosen randomly to be either 1, 2, or $3ms$, and temporal parameters $\tau_n$ chosen as $40ms$ for excitatory neurons and $20ms$ for inhibitory ones. Turning external input currents off after a while we obtained sustained firing activities in the range of 100-150 Hz. We have found, in addition to synchronous and asynchronous realizations of this attractor, a mode of *distributed synchrony*. A characteristic example of a long cycle is shown in Figure 2: The 100 excitatory neurons split into groups such that each group fires at the same frequency and at a fixed phase difference from any other group. The $J_{ij}^{EE}$ synaptic efficacies

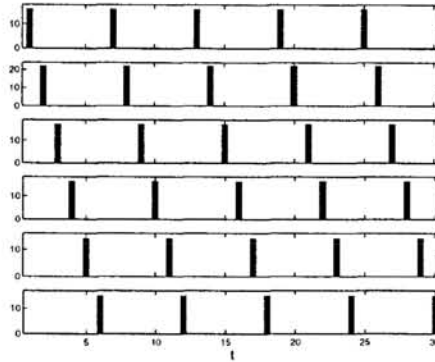

Figure 2: Distributed synchronized firing mode. The firing patterns of six cell assemblies of excitatory neurons are displayed *vs* time (in ms). These six groups of neurons formed in a self-organized manner for a kernel function with equal potentiation and depression. The delays were chosen randomly from three values, 1 2 or 3ms, and the system is monitored every 0.5ms.

are initiated as small random values. The learning process leads to the self-organized synaptic matrix displayed in Figure 3(a). The block form of this matrix represents the ordered couplings that are responsible for the fact that each coherent group of neurons feeds the activity of groups that follow it. The self-organized groups form spontaneously. When the synapses are affected by some external noise, as can come about from Hebbian learning in which these neurons are being coupled with other pools of neurons, the groups will change and regroup, as seen in Figure 3(b) and 3(c).

(a)                              (b)                              (c)

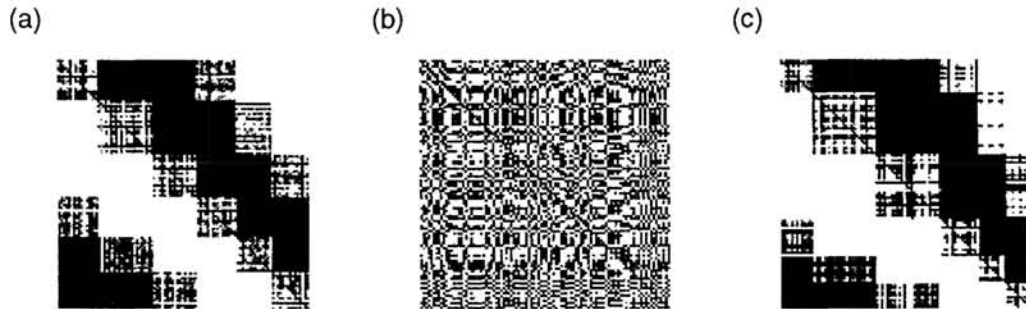

Figure 3: A synaptic matrix for $n = 6$ distributed synchrony. The synaptic matrix between the 100 excitatory neurons of our system is displayed in a grey-level code with black meaning zero efficacy and white standing for the synaptic upper-bound. **(a)** The matrix that exists during the distributed synchronous mode of Figure 2. Its basis is ordered such that neurons that fire together are grouped together. **(b)** Using the same basis as in (a) a new synaptic matrix is shown, one that is formed after stopping the sustained activity of Figure 2, introducing noise in the synaptic matrix, and reinstituting the original memory training. **(c)** The same matrix as (b) is shown in a new basis that exhibits connections that lead to a new and different realization of distributed synchrony.

A stable distributed synchrony cycle can be simply understood for the case of a single synaptic delay setting the basic step, or phase difference, of the cycle. When several delay parameters exist, a situation that probably more accurately represents the $\alpha$−function character of synaptic transmission in cortical networks, distributed

synchrony may still be obtained, as is evident from Figure 2. After some time the cycle may destabilize and regrouping may occur by itself, without external interference. The likelihood of this scenario is increased because different synaptic connections that have different delays can interfere with one another. Nonetheless, over time scales of the type shown in Figure 2, grouping is stable.

## 4 Analysis of a Cycle

In this section we analyze the dynamics of the network when it is in a stable state of distributed synchrony. We assume that $n$ groups of neurons are formed and calculate the stationary distribution of $J_{ij}^{EE}(t)$. In this state the firing pattern of every two neurons in the network can be characterized by their frequency $\nu(t)$ and by their relative phase $\delta$. We assume that $\delta$ is a random normal variable with mean $\mu_\delta$ and standard deviation $\sigma_\delta$. Thus, Eq. 4 can be rewritten as the following stochastic differential equation

$$dJ_{ij}^{EE}(t) = \left[ \mu_{F_{ij}}(t) - \frac{1}{\tau_s} J_{ij}^{EE}(t) \right] dt + \sigma_{F_{ij}}(t) dW(t) \qquad (7)$$

where $F_{ij}(t)$ (Eq. 5) is represented here by a drift term $\mu_{F_{ij}}(t)$ and a diffusion term $\sigma_{F_{ij}}(t)$ which are its mean and standard deviation. $W(t)$ describes a Wiener process. Note that both $\mu_{F_{ij}}(t)$ and $\sigma_{F_{ij}}(t)$ are calculated for a specific distribution of $\delta$ and are functions of $\mu_\delta$ and $\sigma_\delta$.

The stochastic process that satisfies Eq. 7 will satisfy the Fokker-Plank equation for the probability distribution $f$ of $J_{ij}^{EE}$,

$$\frac{\partial f(J_{ij}^{EE}, t)}{\partial t} = -\frac{\partial}{\partial J_{ij}^{EE}} \left[ \left( \mu_{F_{ij}}(t) - \frac{1}{\tau_s} J_{ij}^{EE} \right) f(J_{ij}^{EE}, t) \right] + \frac{\sigma_{F_{ij}}^2(t)}{2} \frac{\partial^2 f(J_{ij}^{EE}, t)}{\partial J_{ij}^{EE^2}} \qquad (8)$$

with reflecting boundary conditions imposed by the synaptic bounds, 0 and $J_{max}$. Since we are interested in the stable state of the process we solve the stationary equation. The resulting density function is

$$f(J_{ij}^{EE}, \mu_\delta, \sigma_\delta) = \frac{\mathcal{N}}{\sigma_{F_{ij}}^2(t)} \exp \left[ \frac{1}{\sigma_{F_{ij}}^2(t)} \left( 2\mu_{F_{ij}} J_{ij}^{EE} - \frac{1}{\tau_s} J_{ij}^{EE^2} \right) \right] \qquad (9)$$

where

$$\mathcal{N} = \left[ \int_0^{J_{max}} f(J_{ij}^{EE}, \mu_\delta, \sigma_\delta) dJ_{ij}^{EE} \right]^{-1} \qquad (10)$$

Eq. 9 enables us to calculate the stationary distribution of the synaptic efficacies between the presynaptic neuron $i$ and the post-synaptic neuron $j$ given their frequency $\nu$ and the parameters $\mu_\delta$ and $\sigma_\delta$. An example of a solution for a 3-cycle is shown in Figure 4. In this case all neurons fire with frequency $\nu = (3\tau_d)^{-1}$ and $\mu_\delta$ takes one of the values $-\tau_d, 0, \tau_d$.

Simulation results of a 3-cycle in a network of excitatory and inhibitory integrate-and-fire neurons described in Section 2 are given in Figure 5. As can be seen the results obtained from the analysis match those observed in the simulation.

## 5 Discussion

The interesting experimental observations of synaptic learning curves [Markram *et al.*, 1997, Zhang *et al.*, 1998] have led us to study their implications for the firing patterns of a Hebbian cell assembly. We find that, in addition

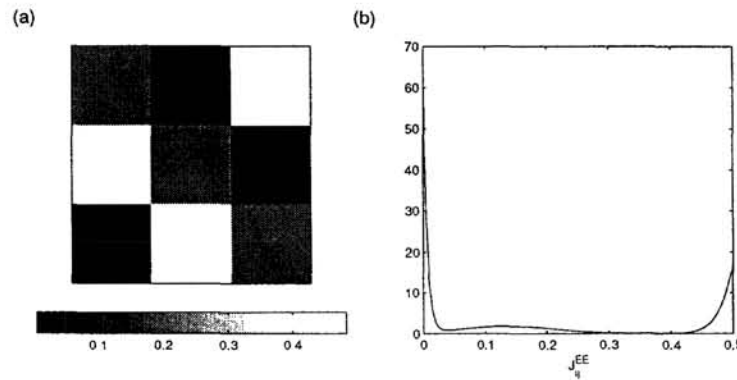

Figure 4: Results of the analysis for $n = 3$, $\sigma_\delta = 2ms$ and $\tau_d = 2.5ms$. (a) The synaptic matrix. Each of the nine blocks symbolizes a group of connections between neurons that have a common phase-lag $\mu_\delta$. The mean of $J_{ij}^{EE}$ was calculated for each cell by Eq. 9 and its value is given by the gray scale tone. (b) The distribution of synaptic values between all excitatory neurons.

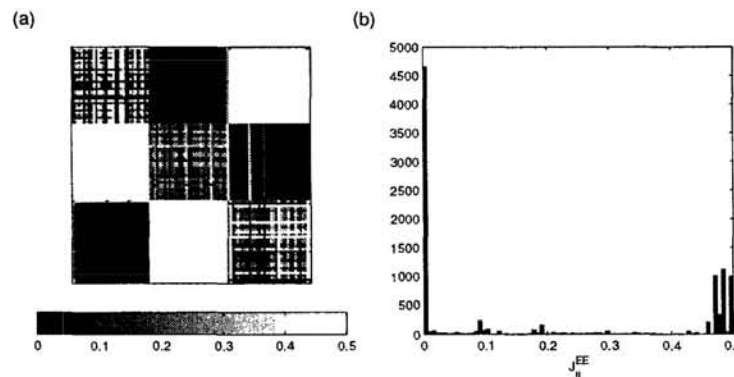

Figure 5: Simulation results for a network of $N_E = 100$ and $N_I = 50$ integrate-and-fire neurons, when the network is in a stable $n = 3$ state. $\tau_n = 10ms$ for both excitatory and inhibitory neurons. The average frequency of the neurons is 130 Hz. (a) The excitatory synaptic matrix. (b) Histogram of the synaptic efficacies.

to the expected synchronous and asynchronous modes, an interesting behavior of distributed synchrony can emerge. This is the phenomenon that we have investigated both by simulations and by analytic evaluation.

Distributed synchrony is a mode in which the Hebbian cell assembly breaks into an $n$-cycle. This cycle is formed by instantaneous symmetry breaking, hence specific classification of neurons into one of the $n$ groups depends on initial conditions, noise, etc. Thus the different groups of a single cycle do not have a semantic invariant meaning of their own. It seems perhaps premature to try and identify these cycles with synfire chains [Abeles, 1982] that show recurrence of firing patterns of groups of neurons with periods of hundreds of ms. Note however, that if we make such an identification, it is a different explanation from the model of [Herrmann *et al.*, 1995], which realizes the synfire chain by combining sets of preexisting patterns into a cycle.

The simulations in Figures 2 and 3 were carried out with a learning curve that possessed equal potentiation and depression branches, i.e. was completely anti-symmetric in its argument. In that case no synaptic decay was allowed. Figure 5, on the other hand, had stronger potentiation than depression, and a finite synaptic

decay time was assumed. Other conditions in these nets were different too, yet both had a window of parameters where distributed synchrony showed up. Using the analytic approach of section 4 we can derive the probability distribution of synaptic values once a definite cyclic pattern of distributed synchrony is formed. An analytic solution of the combined dynamics of both the synapses and the spiking neurons is still an open challenge. Hence we have to rely on the simulations to prove that distributed synchrony is a natural spatiotemporal behavior that follows from combined neuronal dynamics and synaptic learning as outlined in section 2. To the extent that both types of dynamics reflect correctly the dynamics of cortical neural networks, we may expect distributed synchrony to be a mode in which neuronal attractors are being realized.

The mode of distrbuted synchrony is of special significance to the field of neural computation since it forms a bridge between the feedback and feed-forward paradigms. Note that whereas the attractor that is formed by the Hebbian cell assembly is of global feedback nature, i.e. one may regard all neurons of the assembly as being connected to other neurons within the same assembly, the emerging structure of distributed synchrony shows that it breaks down into groups. These groups are connected to one another in a self-organized feed-forward manner, thus forming the cyclic behavior we have observed.

# References

[Abbott and Song, 1999] L. F. Abbott and S. Song. Temporally asymmetric hebbian learning, spike timing and neuronal response variability. In M. S. Kearns, S. A. Solla, and D. A. Cohn, editors, *Advances in Neural Information Processing Systems 11: Proceedings of the 1998 Conference*, pages 69 – 75. MIT Press, 1999.

[Abeles, 1982] M. Abeles. *Local Cortical Circuits*. Springer, Berlin, 1982.

[Brunel, 1999] N. Brunel. Dynamics of sparsely connected networks of excitatory and inhibitory spiking neurons. *Journal of Computational Neuroscience*, 1999.

[Herrmann *et al.*, 1995] M. Herrmann, J. Hertz, and A. Prügel-Bennet. Analysis of synfire chains. *Network: Comp. in Neural Systems*, 6:403 – 414, 1995.

[Kempter *et al.*, 1999] R. Kempter, W. Gerstner, and J. Leo van Hemmen. Spike-based compared to rate-based hebbian learning. In M. S. Kearns, S. A. Solla, and D. A. Cohn, editors, *Advances in Neural Information Processing Systems 11: Proceedings of the 1998 Conference*, pages 125 – 131. MIT Press, 1999.

[Markram *et al.*, 1997] H. Markram, J. Lübke, M. Frotscher, and B. Sakmann. Regulation of synaptic efficacy by coincidence of postsynaptic aps and epsps. *Science*, 275(5297):213 – 215, 1997.

[Miyashita, 1988] Y. Miyashita. Neuronal correlate of visual associative long-term memory in the primate temporal cortex. *Nature*, 335:817 – 820, 1988.

[Yakovlev *et al.*, 1998] V. Yakovlev, S. Fusi, E. Berman, and E. Zohary. Inter-trial neuronal activity in inferior temporal cortex: a putative vehicle to generate long-term visual associations. *Nature Neurosc.*, 1(4):310 – 317, 1998.

[Zhang *et al.*, 1998] L. I. Zhang, H. W. Tao, C. E. Holt, W. A. Harris, and M. Poo. A critical window for cooperation and competition among developing retinotectal synapses. *Nature*, 395:37 – 44, 1998.
